# Bayesian Monte Carlo

**Carl Edward Rasmussen** and **Zoubin Ghahramani**
Gatsby Computational Neuroscience Unit
University College London
17 Queen Square, London WC1N 3AR, England
edward,zoubin@gatsby.ucl.ac.uk
http://www.gatsby.ucl.ac.uk

## Abstract

We investigate Bayesian alternatives to classical Monte Carlo methods for evaluating integrals. Bayesian Monte Carlo (BMC) allows the incorporation of prior knowledge, such as smoothness of the integrand, into the estimation. In a simple problem we show that this outperforms any classical importance sampling method. We also attempt more challenging multidimensional integrals involved in computing marginal likelihoods of statistical models (a.k.a. partition functions and model evidences). We find that Bayesian Monte Carlo outperformed Annealed Importance Sampling, although for very high dimensional problems or problems with massive multimodality BMC may be less adequate. One advantage of the Bayesian approach to Monte Carlo is that samples can be drawn from any distribution. This allows for the possibility of active design of sample points so as to maximise information gain.

## 1  Introduction

Inference in most interesting machine learning algorithms is not computationally tractable, and is solved using approximations. This is particularly true for Bayesian models which require evaluation of complex multidimensional integrals. Both analytical approximations, such as the Laplace approximation and variational methods, and Monte Carlo methods have recently been used widely for Bayesian machine learning problems. It is interesting to note that Monte Carlo itself is a purely frequentist procedure [O'Hagan, 1987; MacKay, 1999]. This leads to several inconsistencies which we review below, outlined in a paper by O'Hagan [1987] with the title "Monte Carlo is Fundamentally Unsound". We then investigate Bayesian counterparts to the classical Monte Carlo.

Consider the evaluation of the integral:

$$\bar{f}_p = \int f(x)p(x)dx, \tag{1}$$

where $p(x)$ is a probability (density), and $f(x)$ is the function we wish to integrate. For example, $p(x)$ could be the posterior distribution and $f(x)$ the predictions made by a model with parameters $x$, or $p(x)$ could be the parameter prior and $f(x) = p(y|x)$ the likelihood so that equation (1) evaluates the marginal likelihood (evidence) for a model. Classical

Monte Carlo makes the approximation:

$$\bar{f}_p \simeq \frac{1}{T} \sum_{t=1}^{T} f(x^{(t)}), \tag{2}$$

where $x^{(t)}$ are random (not necessarily independent) draws from $p(x)$, which converges to the right answer in the limit of large numbers of samples, $T$. If sampling directly from $p(x)$ is hard, or if high density regions in $p(x)$ do not match up with areas where $f(x)$ has large magnitude, it is also possible to draw samples from some *importance sampling distribution* $q(x)$ to obtain the estimate:

$$\bar{f}_p = \int \frac{f(x)p(x)}{q(x)} q(x)\, dx \simeq \frac{1}{T} \sum_t \frac{f(x^{(t)})p(x^{(t)})}{q(x^{(t)})}. \tag{3}$$

As O'Hagan [1987] points out, there are two important objections to these procedures. First, the estimator not only depends on the values of $f(x^{(t)})p(x^{(t)})$ but also on the entirely arbitrary choice of the sampling distribution $q(x)$. Thus, if the same set of samples $\{x^{(1)}, \ldots, x^{(T)}\}$, conveying exactly the same information about $\bar{f}_p$, were obtained from two different sampling distributions, two different estimates of $\bar{f}_p$ would be obtained. This dependence on irrelevant (ancillary) information is unreasonable and violates the *Likelihood Principle*. The second objection is that classical Monte Carlo procedures entirely ignore the values of the $x^{(t)}$ when forming the estimate. Consider the simple example of three points that are sampled from $q$ and the third happens to fall on the same point as the second, $x^{(3)} = x^{(2)}$, conveying no extra information about the integrand. Simply averaging the integrand at these three points, which is the classical Monte Carlo estimate, is clearly inappropriate; it would make much more sense to average the first two (or the first and third). In practice points are unlikely to fall on top of each other in continuous spaces, however, a procedure that weights points equally regardless of their spatial distribution is ignoring relevant information. To summarize the objections, classical Monte Carlo bases its estimate on irrelevant information and throws away relevant information.

We seek to turn the problem of evaluating the integral (1) into a Bayesian inference problem which, as we will see, avoids the inconsistencies of classical Monte Carlo and can result in better estimates. To do this, we think of the unknown desired quantity $\bar{f}_p$ as being random. Although this interpretation is not the most usual one, it is entirely consistent with the Bayesian view that all forms of uncertainty are represented using probabilities: in this case uncertainty arises because we cannot afford to compute $f(x)$ at every location. Since the desired $\bar{f}_p$ is a function of $f(x)$ (which is unknown until we evaluate it) we proceed by putting a prior on $f$, combining it with the observations to obtain the posterior over $f$ which in turn implies a distribution over the desired $\bar{f}_p$.

A very convenient way of putting priors over functions is through Gaussian Processes (GP). Under a GP prior the joint distribution of any (finite) number of function values (indexed by the inputs, $x$) is Gaussian:

$$\mathbf{f} = (f(x^{(1)}), f(x^{(2)}), \ldots, f(x^{(n)}))^{\top} \sim \mathcal{N}(0, K), \tag{4}$$

where here we take the mean to be zero. The covariance matrix is given by the *covariance function*, a convenient choice being:[1]

$$K_{pq} = \mathrm{Cov}(f(x^{(p)}), f(x^{(q)})) = w_0 \exp\left(-\frac{1}{2} \sum_{d=1}^{D} (x_d^{(p)} - x_d^{(q)})^2 / w_d^2\right), \tag{5}$$

where the $w$ parameters are hyperparameters. Gaussian processes, including optimization of hyperparameters, are discussed in detail in [Williams and Rasmussen, 1996].

## 2 The Bayesian Monte Carlo Method

The Bayesian Monte Carlo method starts with a prior over the function, $p(f)$ and makes inferences about $f$ from a set of samples $\mathcal{D} = \{(x^{(i)}, f(x^{(i)}))|i = 1 \ldots n\}$ giving the posterior distribution $p(f|\mathcal{D})$. Under a GP prior the posterior is (an infinite dimensional joint) Gaussian; since the integral eq. (1) is just a linear projection (on the direction defined by $p(x)$), the posterior $p(\bar{f}_p|\mathcal{D})$ is also Gaussian, and fully characterized by its mean and variance. The average over functions of eq. (1) is the expectation of the average function:

$$E_{f|\mathcal{D}}[\bar{f}_p] = \iint f(x)p(x)dx \; p(f|\mathcal{D})df$$
$$= \int \Big[ \int f(x)p(f|\mathcal{D})df \Big] p(x)dx = \int \bar{f}_{\mathcal{D}}(x)p(x)dx, \quad (6)$$

where $\bar{f}_{\mathcal{D}}$ is the posterior mean function. Similarly, for the variance:

$$V_{f|\mathcal{D}}[\bar{f}_p] = \int \Big[ \int f(x)p(x)dx - \int \bar{f}(x')p(x')dx' \Big]^2 p(f|\mathcal{D})df$$
$$= \iiint [f(x) - \bar{f}(x)] [f(x') - \bar{f}(x')] p(f|\mathcal{D})df\, p(x)p(x')dxdx' \quad (7)$$
$$= \iint \mathrm{Cov}_{\mathcal{D}}\big(f(x), f(x')\big)p(x)p(x')dxdx',$$

where $\mathrm{Cov}_{\mathcal{D}}$ is the posterior covariance. The standard results for the GP model for the posterior mean and covariance are:

$$\bar{f}_{\mathcal{D}}(x) = k(x, \mathbf{x})K^{-1}\mathbf{f}, \;\; \text{and} \;\; \mathrm{Cov}_{\mathcal{D}}\big(f(x), f(x')\big) = k(x, x') - k(x, \mathbf{x})K^{-1}k(\mathbf{x}, x'), \;\; (8)$$

where $\mathbf{x}$ and $\mathbf{f}$ are the observed inputs and function values respectively. In general combining eq. (8) with eq. (6-7) may lead to expressions which are difficult to evaluate, but there are several interesting special cases.

If the density $p(x)$ and the covariance function eq. (5) are both Gaussian, we obtain analytical results. In detail, if $p(x) = \mathcal{N}(b, B)$ and the Gaussian kernels on the data points are $\mathcal{N}(a_i = x^{(i)}, A = \mathrm{diag}(w_1^2, \ldots, w_D^2))$ then the expectation evaluates to:

$$E_{f|\mathcal{D}}[\bar{f}_p] = z^\top K^{-1}\mathbf{f}, \;\; z = w_0|A^{-1}B + I|^{-1/2}\exp[-0.5(a - b)^\top(A + B)^{-1}(a - b)] \;\; (9)$$

a result which has previously been derived under the name of Bayes-Hermite Quadrature [O'Hagan, 1991]. For the variance, we get:

$$V_{f|\mathcal{D}}[\bar{f}_p] = w_0 \left| 2A^{-1}B + I \right|^{-1/2} - z^\top K^{-1}z, \quad (10)$$

with $z$ as defined in eq. (9). Other choices that lead to analytical results include polynomial kernels and mixtures of Gaussians for $p(x)$.

### 2.1 A Simple Example

To illustrate the method we evaluated the integral of a one-dimensional function under a Gaussian density (figure 1, left). We generated samples independently from $p(x)$, evaluated $f(x)$ at those points, and optimised the hyperparameters of our Gaussian process fit to the function. Figure 1 (middle) compares the error in the Bayesian Monte Carlo (BMC) estimate of the integral (1) to the Simple Monte Carlo (SMC) estimate using the same samples. As we would expect the squared error in the Simple Monte Carlo estimate decreases as $1/T$ where $T$ is the sample size. In contrast, for more than about 10 samples, the BMC estimate improves at a much higher rate. This is achieved because the prior on $f$ allows

the method to interpolate between sample points. Moreover, whereas the SMC estimate is invariant to permutations of the values on the $x$ axis, BMC makes use of the smoothness of the function. Therefore, a point in a sparse region is far more informative about the shape of the function for BMC than points in already densely sampled areas. In SMC if two samples happen to fall close to each other the function value there will be counted with double weight. This effect means that large numbers of samples are needed to adequately represent $p(x)$. BMC circumvents this problem by analytically integrating its mean function w.r.t. $p(x)$.

In figure 1 left, the negative log density of the true value of the integral under the predictive distributions are compared for BMC and SMC. For not too small sample sizes, BMC outperforms SMC. Notice however, that for very small sample sizes BMC occasionally has very bad performance. This is due to examples where the random draws of $x$ lead to function values $f(x)$ that are consistent with much longer length scale than the true function; the mean prediction becomes somewhat inaccurate, but worse still, the inferred variance becomes very small (because a very slowly varying function is inferred), leading to very poor performance compared to SMC. This problem is to a large extent caused by the *optimization* of the length scale hyperparameters of the covariance function; we ought instead to have integrated over all possible length scales. This integration would effectively "blend in" distributions with much larger variance (since the data is also consistent with a shorter length scale), thus alleviating the problem, but unfortunately this is not possible in closed form. The problem disappears for sample sizes of around 16 or greater.

In the previous example, we chose $p(x)$ to be Gaussian. If you wish to use BMC to integrate w.r.t. non-Gaussian densities then an importance re-weighting trick becomes necessary:

$$\int f(x)p(x)dx = \int \frac{f(x)p(x)}{q(x)}q(x)dx, \tag{11}$$

where the Gaussian process models $f(x)p(x)/q(x)$ and $q(x)$ is a Gaussian and $p(x)$ is an arbitrary density which can be evaluated. See Kennedy [1998] for extension to non-Gaussian $q(x)$.

## 2.2 Optimal Importance Sampler

For the simple example discussed above, it is also interesting to ask whether the efficiency of SMC could be improved by generating independent samples from more-cleverly designed distributions. As we have seen in equation (3), importance sampling gives an unbiased estimate of $\bar{f}_p$ by sampling $x^{(t)}$ from $q(x)$ and computing:

$$\hat{f}_T = \frac{1}{T}\sum_t \frac{f(x^{(t)})p(x^{(t)})}{q(x^{(t)})} \tag{12}$$

where $q(x) > 0$ wherever $p(x) > 0$. The variance of this estimator is given by:

$$V(\hat{f}_T) = \frac{1}{T}\left[\int \frac{f(x)^2 p(x)^2}{q(x)}dx - \bar{f}_p^2\right] \tag{13}$$

Using calculus of variations it is simple to show that the optimal (minimum variance) importance sampling distribution is:

$$q^*(x) = \frac{|f(x)|p(x)}{\int |f(x')|p(x')\,dx'} \tag{14}$$

which we can substitute into equation (13) to get the minimum variance, $V^*$. If $f(x)$ is always non-negative or non-positive then $V^* = 0$, which is unsurprising given that we needed to know $\bar{f}$ in advance to normalise $q$. For functions that take on both positive and

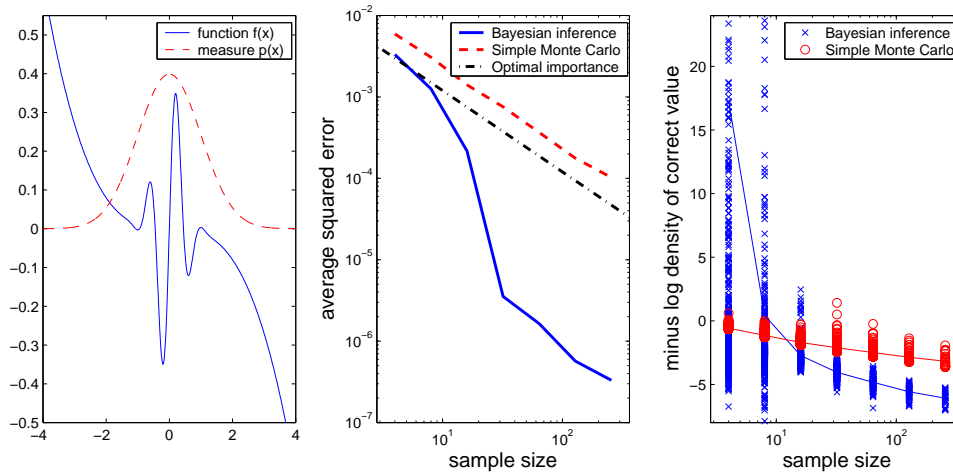

Figure 1: Left: a simple one-dimensional function $f$ (full) and Gaussian density (dashed) with respect to which we wish to integrate $f$. Middle: average squared error for simple Monte Carlo sampling from $p$ (dashed), the optimal achievable bound for importance sampling (dot-dashed), and the Bayesian Monte Carlo estimates. The values plotted are averages over up to 2048 repetitions. Right: Minus the log of the Gaussian predictive density with mean eq. (6) and variance eq. (7), evaluated at the true value of the integral (found by numerical integration), 'x'. Similarly for the Simple Monte Carlo procedure, where the mean and variance of the predictive distribution are computed from the samples, 'o'.

negative values $V^* = (1/T)(E_p[|f(x)|]^2 - \bar{f}^2)$ which is a constant times the variance of a Bernoulli random variable (sign $f(x)$). The lower bound from this optimal importance sampler as a function of number of samples is shown in figure 1, middle. As we can see, Bayesian Monte Carlo improves on the optimal importance sampler considerably. We stress that the optimal importance sampler is not practically achievable since it requires knowledge of the quantity we are trying to estimate.

## 3   Computing Marginal Likelihoods

We now consider the problem of estimating the marginal likelihood of a statistical model. This problem is notoriously difficult and very important, since it allows for comparison of different models. In the physics literature it is known as free-energy estimation. Here we compare the Bayesian Monte Carlo method to two other techniques: Simple Monte Carlo sampling (SMC) and Annealed Importance Sampling (AIS).

Simple Monte Carlo, sampling from the prior, is generally considered inadequate for this problem, because the likelihood is typically sharply peaked and samples from the prior are unlikely to fall in these confined areas, leading to huge variance in the estimates (although they are unbiased). A family of promising "thermodynamic integration" techniques for computing marginal likelihoods are discussed under the name of Bridge and Path sampling in [Gelman and Meng, 1998] and Annealed Importance Sampling (AIS) in [Neal, 2001]. The central idea is to divide one difficult integral into a series of easier ones, parameterised

by (inverse) temperature, $\tau$. In detail:

$$\frac{Z_K}{Z_0} = \frac{Z_1}{Z_0}\frac{Z_2}{Z_1}\cdots\frac{Z_K}{Z_{K-1}}, \quad \text{where}$$

$$Z_0 = \int p(x)dx = 1 \quad \text{and} \quad Z_k = \int p(y|x)^{\tau(k)}p(x)dx, \tag{15}$$

where $\tau(k)$ is the $k^{\text{th}}$ inverse temperature of the annealing schedule and $\tau(K) = 1$. To compute each fraction we sample from equilibrium from the distribution $q_{k-1}(x) \propto p(y|x)^{\tau(k-1)}p(x)$ and compute importance weights:

$$\frac{Z_k}{Z_{k-1}} = \int \frac{p(y|x)^{\tau(k)}p(x)}{p(y|x)^{\tau(k-1)}p(x)}q_{k-1}(x)dx \simeq \frac{1}{T}\sum_{i=1}^{T} p(y|x^{(i)})^{\tau(k)-\tau(k-1)}. \tag{16}$$

In practice $T$ can be set to 1, to allow very slow reduction in temperature. Each of the intermediate ratios are much easier to compute than the original ratio, since the likelihood function to the power of a small number is much better behaved that the likelihood itself. Often elaborate non-linear cooling schedules are used, but for simplicity we will just take a linear schedule for the inverse temperature. The samples at each temperature are drawn using a single Metropolis proposal, where the proposal width is chosen to get a fairly high fraction of acceptances.

The model in question for which we attempt to compute the marginal likelihood was itself a Gaussian process regression fit to the an artificial dataset suggested by [Friedman, 1988].[2] We had 5 length scale hyperparameters, a signal variance ($w_0$) and an explicit noise variance parameter. Thus the marginal likelihood is an integral over a 7 dimensional hyperparameter space. The log of the hyperparameters are given $\mathcal{N}(0, \sigma^2 = 4)$ priors.

Figure 2 shows a comparison of the three methods. Perhaps surprisingly, AIS and SMC are seen to be very comparable, which can be due to several reasons: 1) whereas the SMC samples are drawn independently, the AIS samples have considerable auto-correlation because of the Metropolis generation mechanism, which hampers performance for low sample sizes, 2) the annealing schedule was not optimized nor the proposal width adjusted with temperature, which might possibly have sped up convergence. Further, the difference between AIS and SMC would be more dramatic in higher dimensions and for more highly peaked likelihood functions (i.e. more data).

The Bayesian Monte Carlo method was run on the same samples as were generate by the AIS procedure. Note that BMC can use samples from *any* distribution, as long as $p(x)$ can be evaluated. Another obvious choice for generating samples for BMC would be to use an MCMC method to draw samples from the posterior. Because BMC needs to model the integrand using a GP, we need to limit the number of samples since computation (for fitting hyperparameters and computing the $\alpha$'s) scales as $n^3$. Thus for sample size greater than 2048 we limit the number of samples to 2048, chosen equally spaced from the AIS Markov chain. Despite this thinning of the samples we see a generally superior performance of BMC, especially for smaller sample sizes. In fact, BMC seems to perform equally well for almost any of the investigated sample sizes. Even for this fairly large number of samples, the generation of points from the AIS still dominates compute time.

## 4 Discussion

An important aspect which we have not explored in this paper is the idea that the GP model used to fit the integrand gives errorbars (uncertainties) on the integrand. These error bars

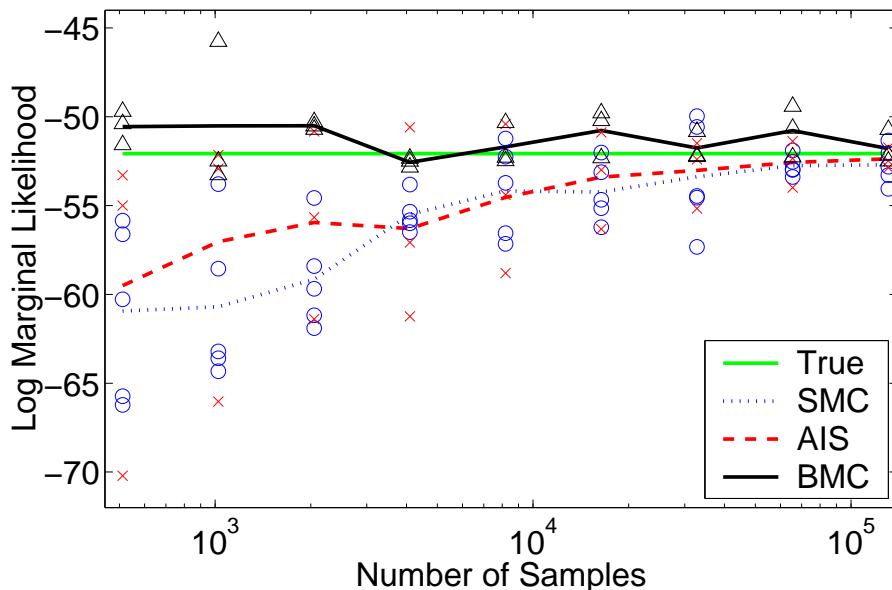

Figure 2: Estimates of the marginal likelihood for different sample sizes using Simple Monte Carlo sampling (SMC; circles, dotted line), Annealed Importance Sampling (AIS; ×, dashed line), and Bayesian Monte Carlo (BMC; triangles, solid line). The true value (solid straight line) is estimated from a single $10^6$ sample long run of AIS. For comparison, the maximum log likelihood is $-37.34$ (which is an upper bound on the true value).

could be used to conduct an experimental design, i.e. active learning. A simple approach would be to evaluate the function at points $x$ where the GP has large uncertainty $\sigma(x)$ and $p(x)$ is not too small: the expected contribution to the uncertainty in the estimate of the integral scales as $\sigma(x)p(x)$. For a fixed Gaussian Process covariance function these design points can often be pre-computed, see e.g. [Minka, 2000]. However, as we are adapting the covariance function depending on the observed function values, active learning would have to be an integral part of the procedure. Classical Monte Carlo approaches cannot make use of active learning since the samples need to be drawn from a given distribution.

When using BMC to compute marginal likelihoods, the Gaussian covariance function used here (equation 5) is not ideally suited to modeling the likelihood. Firstly, likelihoods are non-negative whereas the prior is not restricted in the values the function can take. Secondly, the likelihood tends to have some regions of high magnitude and variability and other regions which are low and flat; this is not well-modelled by a stationary covariance function. In practice this misfit between the GP prior and the function modelled has even occasionally led to negative values for the estimate of the marginal likelihood! There could be several approaches to improving the appropriateness of the prior. An importance distribution such as one computed from a Laplace approximation or a mixture of Gaussians can be used to dampen the variability in the integrand [Kennedy, 1998]. The GP could be used to model the log of the likelihood [Rasmussen, 2002]; however this makes integration more difficult.

The BMC method outlined in this paper can be extended in several ways. Although the choice of Gaussian process priors is computationally convenient in certain circumstances, in general other function approximation priors can be used to model the integrand. For discrete (or mixed) variables the GP model could still be used with appropriate choice of covariance function. However, the resulting sum (analogous to equation 1) may be difficult

to evaluate. For discrete $f$, GPs are not directly applicable.

Although BMC has proven successful on the problems presented here, there are several limitations to the approach. High dimensional integrands can prove difficult to model. In such cases a large number of samples may be required to obtain good estimates of the function. Inference using a Gaussian Process prior is at present limited computationally to a few thousand samples. Further, models such as neural networks and mixture models exhibit an exponentially large number of symmetrical modes in the posterior. Again modelling this with a GP prior would typically be difficult. Finally, the BMC method requires that the distribution $p(x)$ can be evaluated. This contrasts with classical MC where many methods only require that samples can be drawn from some distribution $q(x)$, for which the normalising constant is not necessarily known (such as in equation 16). Unfortunately, this limitation makes it difficult, for example, to design a Bayesian analogue to Annealed Importance Sampling.

We believe that the problem of computing an integral using a limited number of function evaluations should be treated as an inference problem and that all prior knowledge about the function being integrated should be incorporated into the inference. Despite the limitations outlined above, Bayesian Monte Carlo makes it possible to do this inference and can achieve performance equivalent to state-of-the-art classical methods despite using a fraction of sample evaluations, even sometimes exceeding the theoretically optimal performance of some classical methods.

## Acknowledgments

We would like to thank Radford Neal for inspiring discussions.

## Footnotes

[1] Although the function values obtained are assumed to be noise-free, we added a tiny constant to the diagonal of the covariance matrix to improve numerical conditioning.

[2]The data was 100 samples generated from the 5-dimensional function $f(x_1, \ldots, x_5) = 10\sin(\pi x_1 x_2) + 20(x_3 - 0.5)^2 + 10x_4 + 5x_5 + \epsilon$, where $\epsilon$ is zero mean unit variance Gaussian noise and the inputs are sampled independently from a uniform [0, 1] distribution.

## References

Friedman, J. (1988). Multivariate Adaptive Regression Splines. Technical Report No. 102, November 1988, Laboratory for Computational Statistics, Department of Statistics, Stanford University.

Kennedy, M. (1998). Bayesian quadrature with non-normal approximating functions, *Statistics and Computing*, **8**, pp. 365–375.

MacKay, D. J. C. (1999). Introduction to Monte Carlo methods. In *Learning in Graphical Models*, M. I. Jordan (ed), MIT Press, 1999.

Gelman, A. and Meng, X.-L. (1998) Simulating normalizing constants: From importance sampling to bridge sampling to path sampling, *Statistical Science*, vol. 13, pp. 163–185.

Minka, T. P. (2000) Deriving quadrature rules from Gaussian processes, Technical Report, Statistics Department, Carnegie Mellon University.

Neal, R. M. (2001). Annealed Importance Sampling, *Statistics and Computing*, **11**, pp. 125–139.

O'Hagan, A. (1987). Monte Carlo is fundamentally unsound, *The Statistician*, **36**, pp. 247-249.

O'Hagan, A. (1991). Bayes-Hermite Quadrature, *Journal of Statistical Planning and Inference*, **29**, pp. 245–260.

O'Hagan, A. (1992). Some Bayesian Numerical Analysis. *Bayesian Statistics 4* (J. M. Bernardo, J. O. Berger, A. P. Dawid and A. F. M. Smith, eds), Oxford University Press, pp. 345–365 (with discussion).

C. E. Rasmussen (2003). Gaussian Processes to Speed up Hybrid Monte Carlo for Expensive Bayesian Integrals, *Bayesian Statistics 7* (J. M. Bernardo, M. J. Bayarri, J. O. Berger, A. P. Dawid, D. Heckerman, A. F. M. Smith and M. West, eds), Oxford University Press.

Williams, C. K. I. and C. E. Rasmussen (1996). Gaussian Processes for Regression, in D. S. Touretzky, M. C. Mozer and M. E. Hasselmo (editors), NIPS 8, MIT Press.
